# Pairwise Clustering and Graphical Models

**Noam Shental**
Computer Science & Eng.
Center for Neural Computation
Hebrew University of Jerusalem
Jerusalem, Israel 91904
fenoam@cs.huji.ac.il

**Assaf Zomet**
Computer Science & Eng.
Hebrew University of Jerusalem
Jerusalem, Israel 91904
zomet@cs.huji.ac.il

**Tomer Hertz**
Computer Science & Eng.
Center for Neural Computation
Hebrew University of Jerusalem
Jerusalem, Israel 91904
tomboy@cs.huji.ac.il

**Yair Weiss**
Computer Science & Eng.
Center for Neural Computation
Hebrew University of Jerusalem
Jerusalem, Israel 91904
yweiss@cs.huji.ac.il

## Abstract

Significant progress in clustering has been achieved by algorithms that are based on pairwise affinities between the datapoints. In particular, spectral clustering methods have the advantage of being able to divide arbitrarily shaped clusters and are based on efficient eigenvector calculations. However, spectral methods lack a straightforward probabilistic interpretation which makes it difficult to automatically set parameters using training data.

In this paper we use the previously proposed typical cut framework for pairwise clustering. We show an equivalence between calculating the typical cut and inference in an undirected graphical model. We show that for clustering problems with hundreds of datapoints exact inference may still be possible. For more complicated datasets, we show that loopy belief propagation (BP) and generalized belief propagation (GBP) can give excellent results on challenging clustering problems. We also use graphical models to derive a learning algorithm for affinity matrices based on labeled data.

## 1   Introduction

Consider the set of points shown in figure 1a. Datasets of this type, where the two clusters are not easily described by a parametric model can be successfully clustered using pairwise clustering algorithms [4, 6, 3]. These algorithms start by building a graph whose vertices correspond to datapoints and edges exist between nearby points with a weight that decreases with distance. Clustering the points is then equivalent to graph partitioning.

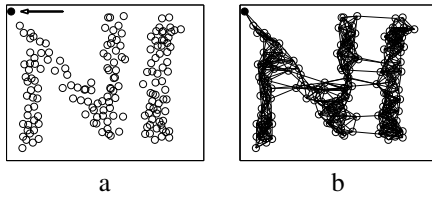

Figure 1: Clustering as graph partitioning (following [8]). Vertices correspond to datapoints and edges between adjacent pixels are weighted by the distance. A single isolated datapoint is marked by an arrow

How would we define a good partitioning? One option is the minimal cut criterion. Define:

$$cut(A, B) = \sum_{i \in A, j \in B} W(i, j) \tag{1}$$

where $W(i, j)$ is the strength of the weight between node i and j in the graph. The minimal cut criterion finds clusterings that minimize $cut(A, B)$.

The advantage of using the minimal cut criterion is that the optimal segmentation can be computed in polynomial time. A disadvantage, pointed out by Shi and Malik [8], is that it will often produce trivial segmentations. Since the cut value grows linearly with the number of edges cut, a single datapoint cut from its neighbors will often have a lower cut value than the desired clustering (e.g, the minimal cut solution separates the full dot in fig1, instead of the desired 'N' and 'I' clusters).

In order to avoid these trivial clusterings, several graph partitioning criteria have been proposed. Shi and Malik suggested the normalized cut criterion which directly penalizes partitions where one of the groups is small, hence a separation of a single isolated datapoint is not favored. Minimization of the normalized cut criterion is NP-Complete but it can be approximated using *spectral methods*.

Despite the success of spectral methods in a wide range of clustering problems, several problems remain. Perhaps the most important one is the lack of a straightforward probabilistic interpretation. However, interesting progress in this direction has be made by Meila and Shi [4] who showed a relation between the top eigenvectors and the equilibrium distribution of a random walk on the graph.

The typical cut criterion, suggested by Blatt et al [1] and later by Gdalyahu et al [2], is based on a simple probabilistic model. Blatt et al first defines a probability distribution over possible partitions by:

$$\Pr(A, B) = \frac{1}{Z} e^{-cut(A, B)/T} \tag{2}$$

where $Z$ is a normalizing constant, and the "temperature" $T$ serves as a free parameter.

Using this probability distribution, the most probable partition is simply the minimal cut. Thus performing MAP inference under this probability distribution will still lead to trivial segmentations. However, as Blatt et al pointed out, there is far more information in the full probability distribution over partitions than solely in the MAP partition. For example, consider the pairwise correlation $p(i, j)$ defined for any two neighboring nodes in the graph as the probability that they belong to the same segment:

$$p(i, j) = \sum_{A, B} \Pr(A, B) SAME(i, j; A, B) \tag{3}$$

with $SAME(i, j; A, B)$ defined as 1 iff $i \in A$ and $j \in A$ or $i \in B$ and $j \in B$.
Referring again, to the single isolated datapoint in figure 1, then while that datapoint and its neighbors do not appear in the same cluster in the most probable partition, they *do* appear in the same cluster for the vast majority of partitions. Thus we would expect $p(i, j) > 1/2$ for that datapoint and its neighbors.

Hence the typical cut algorithm of Blatt et al consists of three stages:

- *Preprocessing*: Construct the affinity matrix $W$ so that each node will be connected to at most $K$ neighbors. Define the affinities $W(i,j)$ as: $W(i,j) = e^{\frac{-d(i,j)^2}{\sigma^2}}$, where $d_{i,j}$ is the distance between points $i$ and $j$, and $\sigma$ is the mean distance to the $K$'th neighbor.
- *Estimating pairwise correlations*: Use a Markov chain Monte-Carlo (MCMC) sampling method to estimate $p(i,j)$ at each temperature $T$.
- *Postprocessing*: Define the *typical cut* partition as the connected components of the graph after removing any links for which $p(i,j) < 1/2$.

For a given $W(i,j)$ the algorithm has a single free temperature parameter $T$ (see eq. 5). This parameter implicitly defines the number of clusters. At zero temperature all the datapoints reside in one cluster (this trivially minimizes the cut value), and at high temperatures every datapoint forms a separate cluster.

In this paper we show that calculating the typical cut is equivalent to performing inference in an undirected graphical model. We use this equivalence to show that in problems with hundreds of datapoints, the typical cut may be calculated exactly. We show that when exact inference is impossible, loopy belief propagation (BP) and generalized belief propagation (GBP) may give an excellent approximation with very little computational cost. Finally, we use the standard algorithm for ML estimation in graphical models to derive a learning algorithm for affinity matrices based on labeled data [1].

## 2  The connection between typical cuts and graphical models

An undirected graphical model with pairwise potentials (see [10] for a review) consists of a graph $G$ and potential functions $\Psi_{ij}(x_i, x_j)$ such that the probability of an assignment $x$ is given by:

$$\Pr(x) = \frac{1}{Z} \prod_{<ij>} \Psi_{ij}(x_i, x_j) \tag{4}$$

where the product is taken over nodes that are connected in the graph $G$.

To connect this to typical cuts we first define for every partition $(A, B)$ a binary vector $x$ such that $x(i) = 0$ if $i \in A$ and $x(i) = 1$ if $i \in B$. We then define:

$$\Psi_{ij}(x_i, x_j) = \begin{pmatrix} 1 & e^{-W(i,j)/T} \\ e^{-W(i,j)/T} & 1 \end{pmatrix} \tag{5}$$

*Observation 1:* The typical cut probability distribution (equation 2) is equivalent to that induced by a pairwise undirected graphical model (equation 4) whose graph $G$ is the same as the graph used for graph partitioning and whose potentials are given by equation 5.

So far we have focused on partitioning the graph into two segments, but the equivalence holds for any number of segments $q$. Let $(A_1, A_2, \cdots, A_q)$ be a partitioning of the graph into $q$ segments (note that these segments need not be connected in $G$). Define $cut(A_1, A_2, \cdots A_q)$ in direct analogy to equation 1, and:

$$\Pr((A_1, A_2, \cdots, A_q)) = \frac{1}{Z} e^{-\frac{1}{T} cut(A_1, A_2, \cdots, A_q)} \tag{6}$$

The implication of observation 1 is that we can use the powerful tools of graphical models in the context of pairwise clustering. In subsequent sections we provide examples of the benefits of using graphical models to compute typical cuts.

# 3 Computing typical cuts using inference in a graphical model

Typical cuts has been successfully used for clustering of datapoints in $R^n$ [1] using an expensive MCMC to calculate pairwise correlations, $p(i,j)$. Using inference algorithms we provide a deterministic and more efficient estimate of $p(i,j)$. More specifically, we use inference algorithms to compute the pairwise beliefs over neighboring nodes $b_{ij}(x_i, x_j)$, and calculate the pairwise correlation as $p(i,j) = \sum_{t=1}^{q} b_{ij}(t,t)$.

In cases where the maximal clique size is small enough, we can calculate $p(i,j)$ *exactly* using the junction tree algorithm. In all other cases we must resort to approximate inference using the BP and the GBP algorithms. The following subsections discuss exact and approximate inference for computing typical cuts.

## 3.1 Exact inference for typical cut clustering

The nature of real life clustering problems seems to suggest that exact inference would be intractable due to the clique size of the junction tree. Surprisingly, in our empirical studies, we discovered that on many datasets, including benchmark problems from the UCI repository, we obtain "thin" junction trees (with maximal clique size less than 20). Figure 2a shows a two dimensional representative result. The temperature parameter $T$ was automatically chosen to provide two large clusters. As shown previously by Gdalyahu et al the typical cut criterion does sensible things: it does not favor segmentation of individual datapoints (as in minimal cut), nor is it fooled by narrow bridges between clusters (as in simple connected components). However, while previous typical cut algorithms *approximate* $p(i,j)$ using MCMC, in some cases using the framework of graphical model we can calculate $p(i,j)$ *exactly* and efficiently.

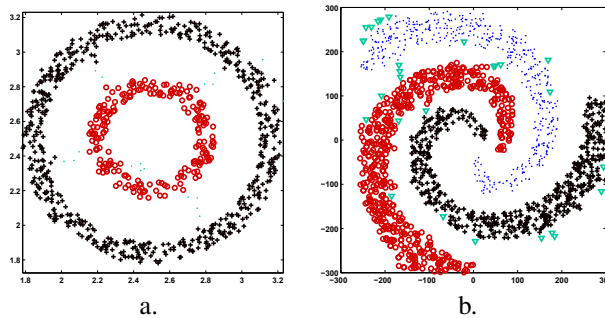

a.            b.

Figure 2: Clustering examples with clusters indicated by different markers. In example (a) the pairwise correlations were calculated *exactly*, while in example (b) we used BP.

## 3.2 Approximate inference for typical cut clustering

Although exact inference is shown to be possible, in the more common case it is infeasible, and $p(i,j)$ can only be estimated using approximate inference algorithms. In this section we discuss approximate inference using the BP and the GBP algorithms.

**Approximate inference using Belief Propagation**    In BP the pairwise beliefs over neighboring nodes, $b_{ij}$, are defined using the messages as:

$$b_{ij}(x_i, x_j) = \alpha \Psi_{ij}(x_i, x_j) \prod_{x_k \in N(x_i) \backslash x_j} m_{ki}(x_i) \prod_{x_k \in N(x_j) \backslash x_i} m_{kj}(x_j) \tag{7}$$

Can this be used as an approximation for pairwise clustering?

*Observation 2:* In case where the messages are initialized uniformly the pairwise beliefs calculated by BP are only a function of the local potentials, i.e $b_{ij}(x_i, x_j) \propto \psi_{ij}(x_i, x_j)$.

*Proof:* Due to the symmetry of the potentials and since the messages are initialized uniformly, all the messages in BP remain uniform. Thus equation 7 will simply give the normalized local potentials.

A consequence of observation 2 is that we need to break the symmetry of the problem in order to use BP. We use here the method of conditioning. Due to the symmetry of the potentials, if exact inference is used then conditioning on a single node $x_c = 1$ and calculating conditional correlations $P(x_i = x_j | x_c = 1)$ should give exactly the same answer as the unconditional correlations $p(i, j) = P(x_i = x_j)$. However, when BP inference is used, clamping the value of $x_c$ causes its outgoing messages to be nonuniform, and as these messages propagate through the graph they break the symmetry used in the proof of observation 2. Empirically, this yields much better approximations of the correlations. In some cases (e.g. when the graph is disconnected) conditioning on a single point does not break the symmetry throughout the graph and additional points need to be clamped.

In order to evaluate the quality of the approximation provided by BP, we compared BP using conditioning and exact inference over the dataset shown in fig 2a. Figure 3 displays the results at two different temperatures: "low" and "high". Each row presents the clustering solution of exact inference and BP, and a scatter plot of the correlations over all of the edges using the two methods. At the "low" temperature the approximation almost coincides with the exact values, but at the "high" temperature BP over estimates the correlation values.

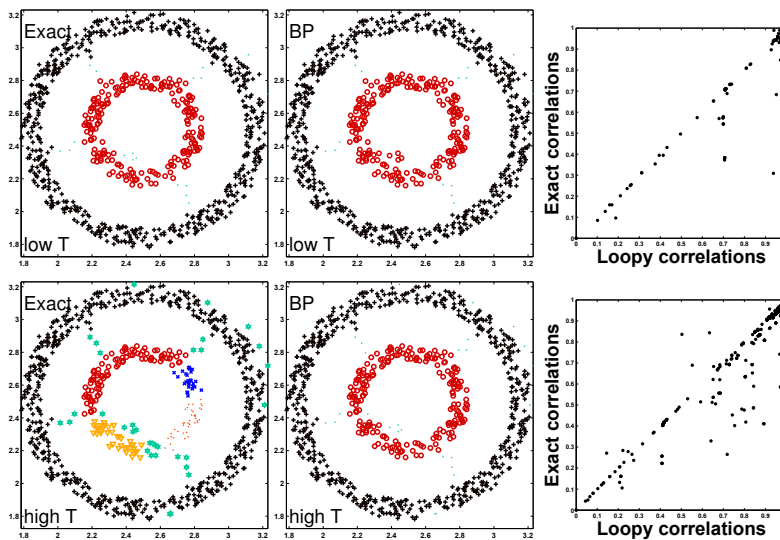

Figure 3: Clustering results at a "low" temperature (upper row) and a "high" temperature (lower row). The left and middle columns present clustering results of exact inference and of BP, respectively. The right column compares the values of the correlations provided by the two methods. Each dot corresponds to an edge in the graph. At "low" temperature most of the correlations are close to 1, hence many edges appear as a single dot.

**Approximate inference using Generalized Belief Propagation** Generalized Belief Propagation algorithms (GBP) [10] extend the BP algorithm by sending messages that are functions of clusters of variables, and has been shown to provide a better approximation than BP in many applications. Can GBP improve the approximation of pairwise correlations in typical cuts?

Our empirical studies show that the performance and convergence of GBP over a general graph obtained from arbitrary points in $R^n$, strongly depends on the initial choice of clusters (regions). As also observed by Minka et al [5] a specific choice of clusters may yield worse results than BP, or may even cause GBP not to converge. However it is far from obvious how to choose these clusters. In previous uses of GBP [10] the basic clusters used were chosen by hand. In order to use GBP to approximate $p(i,j)$ in a general graph, one must obtain a useful *automatic* procedure for selecting these initial clusters. We have experimented with various heuristics but none of them gave good performance. However, in the case of ordered graphs such as 2D grids and images, we have found that GBP gives an excellent approximation when using four neighboring grid points as a region.

Figure 4a shows results of GBP approximations for a $30x30$ 2D uniform grid. The clique size in a junction tree is of order $2^{30}$ hence exact inference is infeasible. We compare the correlations $p(i,j)$ calculated using an extensive MCMC sampling procedure [9] to those calculated using GBP with the clusters being four neighboring pixels in the graph. GBP converges in only 10 iterations and can be seen to provide an excellent approximation.

Figure 4c presents a comparison of the MCMC correlations with those calculated by GBP on a real $120x80$ image shown in fig 4b with affinity based on color similarity. Figure 4d presents the clustering results, which provides a *segmentation* of the image.

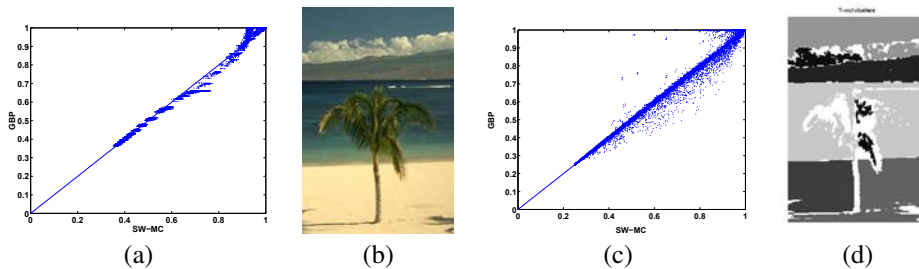

| (a) | (b) | (c) | (d) |

Figure 4: (a) Scatter plot of pairwise correlations in a $30x30$ grid, using MCMC [9] and GBP. Each dot corresponds to the pairwise correlation of one edge at a specific temperature. Notice the excellent correspondence between GBP and MCMC (c) The same comparison performed over the image in (b). (d) shows a gray level map of the 15 largest clusters.

## 4  Learning Affinity Matrices from Labeled Datasets

As noted in the introduction using graphical models to compute typical cuts, can also be advantageous for other aspects of the clustering problem, apart from computing $p(i,j)$. One such important advantage is learning the affinity matrix $W(i,j)$ from labeled data.

In many problems, there are multiple ways to define affinities between any two datapoints. For example, in image segmentation where the nodes are pixels, one can define affinity based on color similarity, texture similarity or some combination of the two. Our goal is to use a labeled training set of manually segmented images to learn the "right" affinities.

More specifically let us assume the "correct" affinity is a linear combination of a set of known affinity functions $\{f_k\}_{k=1}^K$, each corresponding to different features of the data. Hence the affinity between neighboring points $i$ and $j$, is defined by: $W(i,j) = \sum_{k=1}^K \alpha_k f_k(i,j)$. In addition assume we are given a labeled training sample, which consists of the following: (i) A graph in which neighboring nodes are connected by edges. (ii) Affinity values $f_k(i,j)$. (iii) A partition of the graph $x$. Our goal is to estimate the affinity mixing coefficients $\alpha_k$.

This problem can be solved using the graphical model defined by the typical cut probability distribution (Equation 6). Recall that the probability of a partition $x$ is defined as

$$P(x) = \frac{1}{Z}e^{-cut(x)} = \frac{1}{Z}e^{-\sum_{<ij>}(1-\delta(x_i-x_j))W(i,j)} = \frac{1}{Z(\alpha)}e^{-\sum_{k=1}^{K}\alpha_k\mathsf{fcut}_k(x)} \qquad (8)$$

Where we have defined: $\mathsf{fcut}_k(x) = \sum_{<ij>}(1 - \delta(x_i - x_j))f_k(i,j)$. $\mathsf{fcut}_k(x)$ is the cut value defined by $x$ when only taking into account the affinity function $f_k$, hence it can be computed using the training sample. Differentiating the log likelihood with respect to $\alpha_k$ gives the exponential family equation:

$$\frac{\partial \ln P(x)}{\partial \alpha_k} = -\mathsf{fcut}_k(x) + < \mathsf{fcut}_k >_\alpha \qquad (9)$$

Equation 9 gives an intuitive definition for the optimal $\alpha$: the optimal $\alpha$ is the one for which $< \mathsf{fcut}_k >_\alpha = \mathsf{fcut}_k(x)$, i.e, for optimal $\alpha$ the expected values of the cuts for each feature separately, match exactly the values of these cuts in the training set.

Since we are dealing with the exponential family, the likelihood is convex and the ML solution can be found using gradient ascent. To calculate the gradient explicitly, we use the linearity of expectation:

$$< \mathsf{fcut}_k >_\alpha = \sum_{<ij>} < (1 - \delta(y_i - y_j) >_\alpha f_k(i,j) = \sum_{<ij>} (1 - p(i,j)_\alpha)f_k(i,j)$$

Where $p(i,j)_\alpha$ are the pairwise correlations for given values of $\alpha$.

Equation 9 is visually similar to the learning rule derived by Meila and Shi [4] but the cost function they are minimizing is actually different, hence the expectations are taken with respect to completely different distributions.

### 4.1 Combining learning and GBP approximate inference

We experimented with the learning algorithm on images, with the pixels grid as the graph and using GBP for approximating $p(i,j)_\alpha$. The three pixel affinity functions, $\{f_k\}_{k=1}^3$, correspond to the intensity differences in the $R, G, B$ color channels. We used a standard transformation of intensity difference to an affinity function by a Gaussian kernel.

The *left* pane in Fig 5 shows a synthetic example. There is one training image (fig 5a) but two different manual segmentations (fig 5b,c). The first and second training segmentations are based on an illumination-covariant and an illumination-invariant affinities, respectively. We used gradient ascent as given by equation 9. Figure 5d shows a novel image and figures 5e,f show two different pairwise correlations of this image using the learned $\alpha$. Indeed, the algorithm learns to either ignore or not ignore illumination, based on the training set.
The *right* pane in figure 5 shows results on real images. For real images, we found that a preprocessing of the image colors is required in order to learn shadow-invariant linear transformation. This was done by saturating the image colors. The training segmentation (figures 5a,b,c) ignores shadows. On the novel image (figure 5d) the most salient edge is a shadow on the face. Nevertheless, the segmentation based on the learned affinity (figure 5e) ignores the shadows and segments the facial features from each other. In contrast, a typical cut segmentation which uses a naive affinity function (combining the three color channels with uniform weights) segments mostly based on shadows (figure 5f).

## 5 Discussion

Pairwise clustering algorithms have a wide range of applicability due to their ability to find clusters with arbitrary shapes. In this paper we have shown how pairwise clustering can be

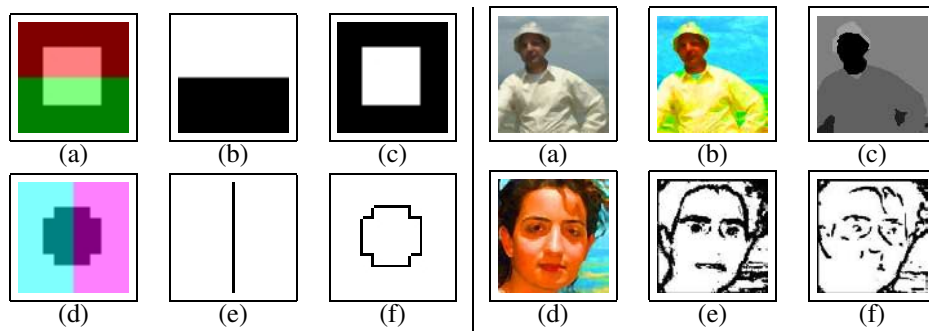

|       |       |       |       |       |       |
|-------|-------|-------|-------|-------|-------|
| (a)   | (b)   | (c)   | (a)   | (b)   | (c)   |
| (d)   | (e)   | (f)   | (d)   | (e)   | (f)   |

Figure 5: **Left pane:** A synthetic example for learning the affinity function. The top row presents the training set: The input image (a), the clusters of the first (b) and second (c) experiments. The bottom row presents the result of the learning algorithm: The input image (d), the marginal probabilities $p(i, j)$ (Eqn. 3) in the first (e) and second (f) experiments. **Right pane:** Learning a color affinity function which is invariant to shadows. The top row shows the learning data set: The input image(a), the pre-processed image (b) and the manual segmentation (invariant to shadows) (c). The bottom row presents, from left to right, the pre-processed test image (d), an edge map produced by learning the shadow-invariant affinity (e) and an edge map produced by a naive affinity function, combining the 3 color channels with uniform weights (f). The edge maps were computed by thresholding the pairwise correlations p(i,j) (Eqn. 3). See text for details. Both illustrations are better viewed in color.

mapped to an inference problem in a graphical model. This equivalence allowed us to use the standard tools of graphical models: exact and approximate inference and ML learning. We showed how to combine approximate inference and ML learning in the challenging problem of learning affinities for images from labeled data. We have only begun to use the many tools of graphical models. We are currently working on learning from unlabeled sets and on other approximate inference algorithms.

## Footnotes

[1]Parts of this work appeared previously in [7].

## References

[1] M. Blatt, S. Wiseman, and E. Domany. Data clustering using a mode lgranular magnet. *Neural Computation*, 9:1805–1842, 1997.

[2] Y. Gdalyahu, D. Weinshall, and M. Werman. Self organization in vision: Stochastic clustering for image segmentation, perceptual grouping, and image database organization. *IEEE Trans. on Pattern Analysis and Machine Intelligence*, 23(10):1053–1074, 2001.

[3] T. Hofmann and J. M. Buhmann. Pairwise data clustering by deterministic annealing. *IEEE Transactions on Pattern Analysis and Machine Intelligence*, 19(1):1–14, 1997.

[4] M. Meila and J. Shi. Learning segmentation by random walks. In *Advances in Neural Information Processing Systems 14*, 2001.

[5] T. Minka and Y. Qi. Tree-structured approximations by expectation propagation. In *Advances in Neural Information Processing Systems 16*, 2003.

[6] A. Ng, M. Jordan, and Y. Weiss. On spectral clustering: Analysis and an algorithm. In *Advances in Neural Information Processing 14*, 2001.

[7] N. Shental, A. Zomet, T. Hertz, and Y. Weiss. Learning and inferring image segmentations using the gbp typical cut. In *9th International Conference on Computer Vision*, 2003.

[8] J. Shi and J. Malik. Normalized cuts and image segmentation. In *Proc. IEEE Conf. Computer Vision and Pattern Recognition*, pages 731–737, 1997.

[9] J.S. Wang and R.H Swendsen. Cluster monte carlo algorithms. *Physica A*, 167:565–579, 1990.

[10] J. Yedidia, W. Freeman, and Y. Weiss. Understanding belief propagation and its generalizations. In G. Lakemeyer and B. Nebel, editors, *Exploring Artificial Intelligence in the New Millennium*. Morgan Kaufmann, 2003.
